# Data Visualization and Feature Selection: New Algorithms for Nongaussian Data

**Howard Hua Yang and John Moody**
Oregon Graduate Institute of Science and Technology
20000 NW, Walker Rd., Beaverton, OR97006, USA
hyang@ece.ogi.edu, moody@cse.ogi.edu, FAX:503 7481406

## Abstract

Data visualization and feature selection methods are proposed based on the *joint mutual information* and ICA. The visualization methods can find many good 2-D projections for high dimensional data interpretation, which cannot be easily found by the other existing methods. The new variable selection method is found to be better in eliminating redundancy in the inputs than other methods based on simple mutual information. The efficacy of the methods is illustrated on a radar signal analysis problem to find 2-D viewing coordinates for data visualization and to select inputs for a neural network classifier.
**Keywords:** feature selection, joint mutual information, ICA, visualization, classification.

## 1 INTRODUCTION

Visualization of input data and feature selection are intimately related. A good feature selection algorithm can identify meaningful coordinate projections for low dimensional data visualization. Conversely, a good visualization technique can suggest meaningful features to include in a model.

Input variable selection is the most important step in the model selection process. Given a target variable, a set of input variables can be selected as explanatory variables by some prior knowledge. However, many irrelevant input variables cannot be ruled out by the prior knowledge. Too many input variables irrelevant to the target variable will not only severely complicate the model selection/estimation process but also damage the performance of the final model.

Selecting input variables after model specification is a model-dependent approach[6]. However, these methods can be very slow if the model space is large. To reduce the computational burden in the estimation and selection processes, we need model-independent approaches to select input variables before model specification. One such approach is $\delta$-Test [7]. Other approaches are based on the *mutual information* (MI) [2, 3, 4] which is very effective in evaluating the relevance of each input variable, but it fails to eliminate redundant variables.

In this paper, we focus on the model-independent approach for input variable selec-

tion based on joint mutual information (JMI). The increment from MI to JMI is the conditional MI. Although the conditional MI was used in [4] to show the monotonic property of the MI, it was not used for input selection.

Data visualization is very important for human to understand the structural relations among variables in a system. It is also a critical step to eliminate some unrealistic models. We give two methods for data visualization. One is based on the JMI and another is based on Independent Component Analysis (ICA). Both methods perform better than some existing methods such as the methods based on PCA and canonical correlation analysis (CCA) for nongaussian data.

## 2   Joint mutual information for input/feature selection

Let $Y$ be a target variable and $X_i$'s are inputs. The relevance of a single input is measured by the MI
$$I(X_i; Y) = K(p(x_i, y) \| p(x_i) p(y))$$
where $K(p\|q)$ is the Kullback-Leibler divergence of two probability functions $p$ and $q$ defined by $K(p(x)\|q(x)) = \sum_x p(x) \log \frac{p(x)}{q(x)}$.

The relevance of a set of inputs is defined by the *joint mutual information*
$$I(X_i, \cdots, X_k; Y) = K(p(x_i, \cdots, x_k, y) \| p(x_i, \cdots, x_k) p(y)).$$

Given two selected inputs $x_j$ and $x_k$, the conditional MI is defined by
$$I(X_i; Y | X_j, X_k) = \sum_{x_j, x_k} p(x_j, x_k) K(p(x_i, y | x_j, x_k) \| p(x_i | x_j, x_k) p(y | x_j, x_k)).$$

Similarly define $I(X_i; Y | X_j, \cdots, X_k)$ conditioned on more than two variables.

The conditional MI is always non-negative since it is a weighted average of the Kullback-Leibler divergence. It has the following property
$$I(X_1, \cdots, X_{n-1}, X_n; Y) - I(X_1, \cdots, X_{n-1}; Y) = I(X_n; Y | X_1, \cdots, X_{n-1}) \geq 0.$$

Therefore, $I(X_1, \cdots, X_{n-1}, X_n; Y) \geq I(X_1, \cdots, X_{n-1}; Y)$, i.e., adding the variable $X_n$ will always increase the mutual information. The information gained by adding a variable is measured by the conditional MI.

When $X_n$ and $Y$ are conditionally independent given $X_1, \cdots, X_{n-1}$, the conditional MI between $X_n$ and $Y$ is
$$I(X_n; Y | X_1, \cdots, X_{n-1}) = 0, \tag{1}$$
so $X_n$ provides no extra information about $Y$ when $X_1, \cdots, X_{n-1}$ are known. In particular, when $X_n$ is a function of $X_1, \cdots, X_{n-1}$, the equality (1) holds. This is the reason why the joint MI can be used to eliminate redundant inputs.

The conditional MI is useful when the input variables cannot be distinguished by the mutual information $I(X_i; Y)$. For example, assume $I(X_1; Y) = I(X_2; Y) = I(X_3; Y)$, and the problem is to select $(x_1, x_2), (x_1, x_3)$ or $(x_2, x_3)$. Since
$$I(X_1, X_2; Y) - I(X_1, X_3; Y) = I(X_2; Y | X_1) - I(X_3; Y | X_1),$$
we should choose $(x_1, x_2)$ rather than $(x_1, x_3)$ if $I(X_2; Y | X_1) > I(X_3; Y | X_1)$. Otherwise, we should choose $(x_1, x_3)$. All possible comparisons are represented by a binary tree in Figure 1.

To estimate $I(X_1, \cdots, X_k; Y)$, we need to estimate the joint probability $p(x_1, \cdots, x_k, y)$. This suffers from the curse of dimensionality when $k$ is large.

Sometimes, we may not be able to estimate high dimensional MI due to the sample shortage. Further work is needed to estimate high dimensional joint MI based on parametric and non-parametric density estimations, when the sample size is not large enough.

In some real world problems such as mining large data bases and radar pulse classification, the sample size is large. Since the parametric densities for the underlying distributions are unknown, it is better to use non-parametric methods such as histograms to estimate the joint probability and the joint MI to avoid the risk of specifying a wrong or too complicated model for the true density function.

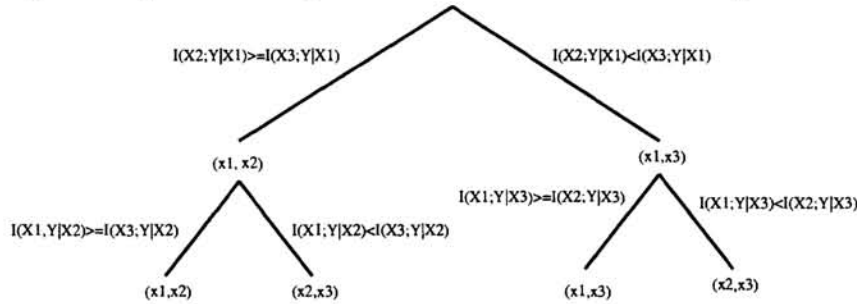

Figure 1: Input selection based on the conditional MI.

In this paper, we use the joint mutual information $I(X_i, X_j; Y)$ instead of the mutual information $I(X_i; Y)$ to select inputs for a neural network classifier. Another application is to select two inputs most relevant to the target variable for data visualization.

## 3   Data visualization methods

We present supervised data visualization methods based on joint MI and discuss unsupervised methods based on ICA.

The most natural way to visualize high-dimensional input patterns is to display them using two of the existing coordinates, where each coordinate corresponds to one input variable. Those inputs which are most relevant to the target variable corresponds the best coordinates for data visualization. Let $(i^*, j^*) = \arg\max_{(i,j)} I(X_i, X_j; Y)$. Then, the coordinate axes $(x_{i^*}, x_{j^*})$ should be used for visualizing the input patterns since the corresponding inputs achieve the maximum joint MI. To find the maximum $I(X_{i^*}, X_{j^*}|Y)$, we need to evaluate every joint MI $I(X_i, X_j; Y)$ for $i < j$. The number of evaluations is $O(n^2)$.

Noticing that $I(X_i, X_j; Y) = I(X_i; Y) + I(X_j; Y|X_i)$, we can first maximize the MI $I(X_i; Y)$, then maximize the conditional MI. This algorithm is suboptimal, but only requires $n - 1$ evaluations of the joint MIs. Sometimes, this is equivalent to exhaustive search. One such example is given in next section.

Some existing methods to visualize high-dimensional patterns are based on dimensionality reduction methods such as PCA and CCA to find the new coordinates to display the data. The new coordinates found by PCA and CCA are orthogonal in Euclidean space and the space with Mahalanobis inner product, respectively. However, these two methods are not suitable for visualizing nongaussian data because the projections on the PCA or CCA coordinates are not statistically independent for nongaussian vectors. Since the JMI method is model-independent, it is better for analyzing nongaussian data.

Both CCA and maximum joint MI are supervised methods while the PCA method is unsupervised. An alternative to these methods is ICA for visualizing clusters [5]. The ICA is a technique to transform a set of variables into a new set of variables, so that statistical dependency among the transformed variables is minimized. The version of ICA that we use here is based on the algorithms in [1, 8]. It discovers a non-orthogonal basis that minimizes mutual information between projections on basis vectors. We shall compare these methods in a real world application.

## 4   Application to Signal Visualization and Classification

### 4.1   Joint mutual information and visualization of radar pulse patterns

Our goal is to design a classifier for radar pulse recognition. Each radar pulse pattern is a 15-dimensional vector. We first compute the joint MIs, then use them to select inputs for the visualization and classification of radar pulse patterns.

A set of radar pulse patterns is denoted by $D = \{(x^i, y^i) : i = 1, \cdots, N\}$ which consists of patterns in three different classes. Here, each $x^i \in R^{15}$ and each $y^i \in \{1, 2, 3\}$.

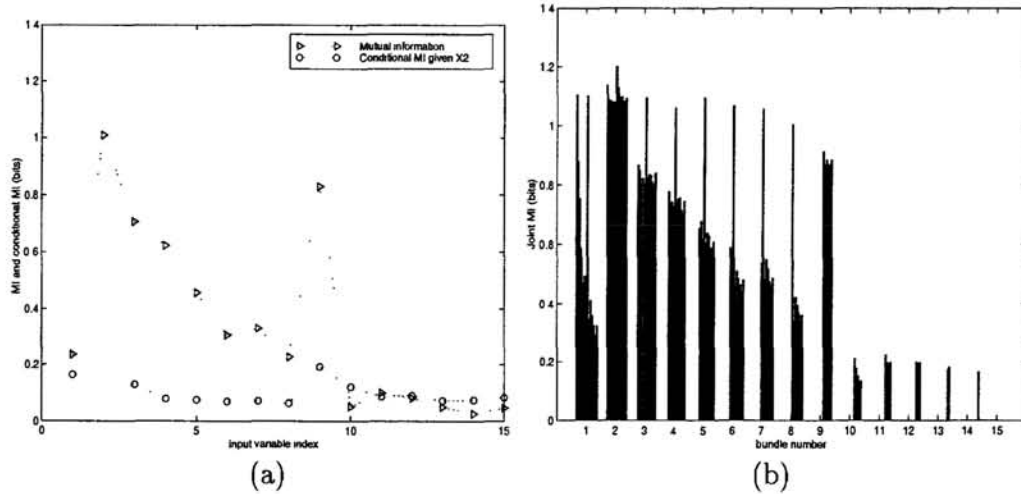

(a)                                                           (b)

Figure 2: (a) MI vs conditional MI for the radar pulse data; maximizing the MI then the conditional MI with $O(n)$ evaluations gives $I(X_{i_1}, X_{j_1}; Y) = 1.201$ bits. (b) The joint MI for the radar pulse data; maximizing the joint MI gives $I(X_{i^*}, X_{j^*}; Y) = 1.201$ bits with $O(n^2)$ evaluations of the joint MI. $(i_1, j_1) = (i^*, j^*)$ in this case.

Let $i_1 = \arg \max_i I(X_i; Y)$ and $j_1 = \arg \max_{j \neq i_1} I(X_j; Y|X_{i_1})$. From Figure 2(a), we obtain $(i_1, j_1) = (2, 9)$ and $I(X_{i_1}, X_{j_1}; Y) = I(X_{i_1}; Y) + I(X_{j_1}; Y|X_{i_1}) = 1.201$ bits. If the number of total inputs is $n$, then the number of evaluations for computing the mutual information $I(X_i; Y)$ and the conditional mutual information $I(X_j; Y|X_{i_1})$ is $O(n)$.

To find the maximum $I(X_{i^*}, X_{j^*}; Y)$, we evaluate every $I(X_i, X_j; Y)$ for $i < j$. These MIs are shown by the bars in Figure 2(b), where the i-th bundle displays the MIs $I(X_i, X_j; Y)$ for $j = i + 1, \cdots, 15$.

In order to compute the joint MIs, the MI and the conditional MI is evaluated $O(n)$ and $O(n^2)$ times respectively. The maximum joint MI is $I(X_{i^*}, X_{j^*}; Y) = 1.201$ bits. Generally, we only know $I(X_{i_1}, X_{j_1}; Y) \leq I(X_{i^*}, X_{j^*}; Y)$. But in this particular

application, the equality holds. This suggests that sometimes we can use an efficient algorithm with only linear complexity to find the optimal coordinate axis view $(x_{i\bullet}, x_{j\bullet})$. The joint MI also gives other good sets of coordinate axis views with high joint MI values.

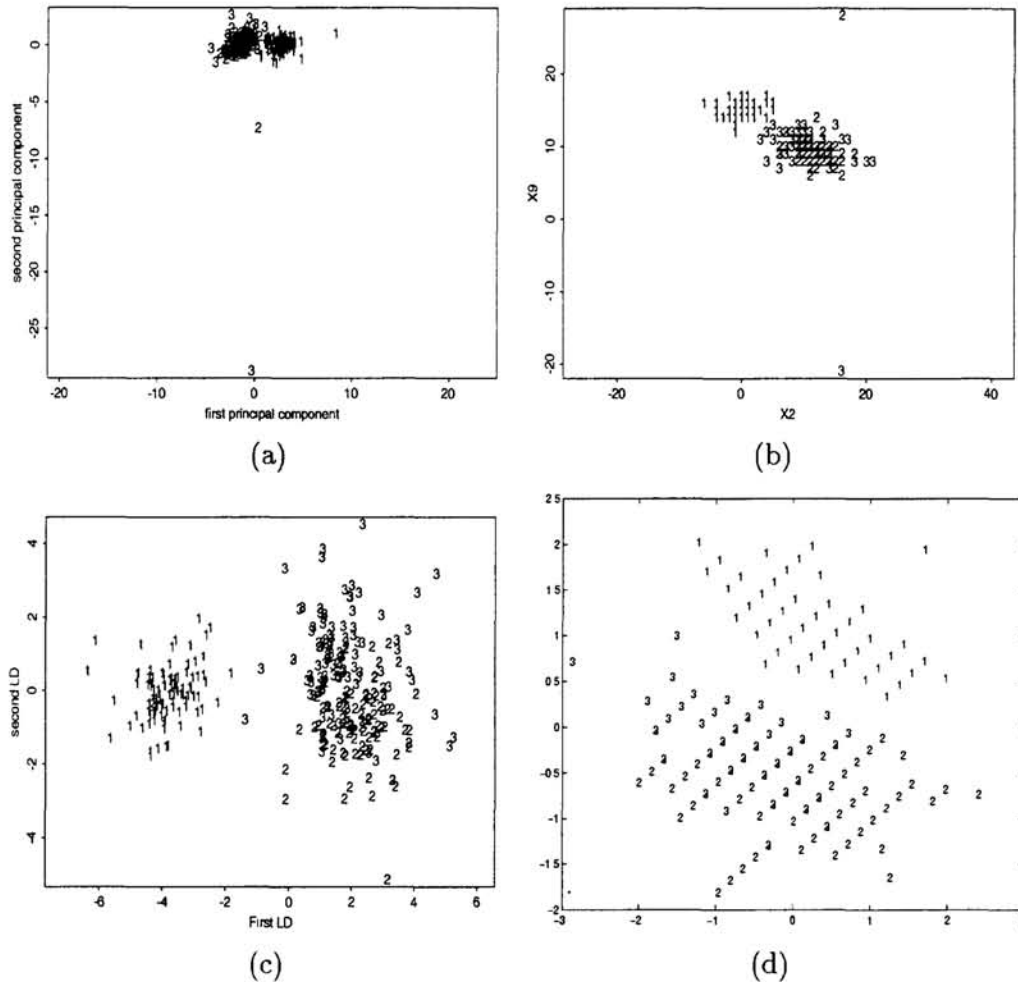

Figure 3: (a) Data visualization by two principal components; the spatial relation between patterns is not clear. (b) Use the optimal coordinate axis view $(x_{i\bullet}, x_{j\bullet})$ found via joint MI to project the radar pulse data; the patterns are well spread to give a better view on the spatial relation between patterns and the boundary between classes. (c) The CCA method. (d) The ICA method.

Each bar in Figure 2(b) is associated with a pair of inputs. Those pairs with high joint MI give good coordinate axis view for data visualization. Figure 3 shows that the data visualizations by the maximum JMI and the ICA is better than those by the PCA and the CCA because the data is nongaussian.

## 4.2 Radar pulse classification

Now we train a two layer feed-forward network to classify the radar pulse patterns. Figure 3 shows that it is very difficult to separate the patterns by using just two inputs. We shall use all inputs or four selected inputs. The data set $D$ is divided

into a training set $D_1$ and a test set $D_2$ consisting of 20 percent patterns in $D$. The network trained on the data set $D_1$ using all input variables is denoted by

$$Y = f(X_1, \cdots, X_n; \boldsymbol{W}_1, \boldsymbol{W}_2, \theta)$$

where $\boldsymbol{W}_1$ and $\boldsymbol{W}_2$ are weight matrices and $\theta$ is a vector of thresholds for the hidden layer.

From the data set $D$, we estimate the mutual information $I(X_i; Y)$ and select $i_1 = \arg \max_i I(X_i; Y)$. Given $X_{i_1}$, we estimate the conditional mutual information $I(X_j; Y | X_{i_1})$ for $j \neq i_1$. Choose three inputs $X_{i_2}, X_{i_3}$ and $X_{i_4}$ with the largest conditional MI. We found a quartet $(i_1, i_2, i_3, i_4) = (1, 2, 3, 9)$. The two-layer feed-forward network trained on $D_1$ with four selected inputs is denoted by

$$Y = g(X_1, X_2, X_3, X_9; \boldsymbol{W}'_1, \boldsymbol{W}'_2, \theta').$$

There are 1365 choices to select 4 input variables out of 15. To set a reference performance for network with four inputs for comparison. Choose 20 quartets from the set $Q = \{(j_1, j_2, j_3, j_4) : 1 \leq j_1 < j_2 < j_3 < j_4 \leq 15\}$. For each quartet $(j_1, j_2, j_3, j_4)$, a two-layer feed-forward network is trained using inputs $(X_{j_1}, X_{j_2}, X_{j_3}, X_{j_4})$. These networks are denoted by

$$Y = h_i(X_{j_1}, X_{j_2}, X_{j_3}, X_{j_4}; \boldsymbol{W}''_1, \boldsymbol{W}''_2, \theta''), \quad i = 1, 2, \cdots, 20.$$

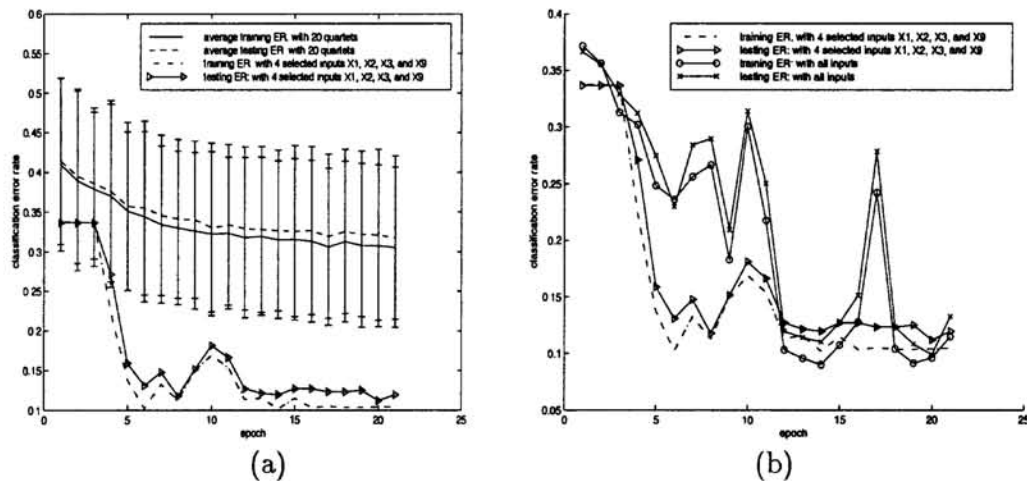

|     |     |
|:---:|:---:|
| (a) | (b) |

Figure 4: (a) The error rates of the network with four inputs $(X_1, X_2, X_3, X_9)$ selected by the joint MI are well below the average error rates (with error bars attached) of the 20 networks with different input quartets randomly selected; this shows that the input quartet $(X_1, X_2, X_3, X_9)$ is rare but informative. (b) The network with the inputs $(X_1, X_2, X_3, X_9)$ converges faster than the network with all inputs. The former uses 65% fewer parameters (weights and thresholds) and 73% fewer inputs than the latter. The classifier with the four best inputs is less expensive to construct and use, in terms of data acquisition costs, training time, and computing costs for real-time application.

The mean and the variance of the error rates of the 20 networks are then computed. All networks have seven hidden units. The training and testing error rates of the networks at each epoch are shown in Figure 4, where we see that the network with four inputs selected by the joint MI performs better than the networks with randomly selected input quartets and converges faster than the network with all inputs. The network with fewer inputs is not only faster in computing but also less expensive in data collection.

## 5 CONCLUSIONS

We have proposed data visualization and feature selection methods based on the *joint mutual information* and ICA.

The maximum JMI method can find many good 2-D projections for visualizing high dimensional data which cannot be easily found by the other existing methods. Both the maximum JMI method and the ICA method are very effective for visualizing nongaussian data.

The variable selection method based on the JMI is found to be better in eliminating redundancy in the inputs than other methods based on simple mutual information. Input selection methods based on mutual information (MI) have been useful in many applications, but they have two disadvantages. First, they cannot distinguish inputs when all of them have the same MI. Second, they cannot eliminate the redundancy in the inputs when one input is a function of other inputs. In contrast, our new input selection method based on the *joint* MI offers significant advantages in these two aspects.

We have successfully applied these methods to visualize radar patterns and to select inputs for a neural network classifier to recognize radar pulses. We found a smaller yet more robust neural network for radar signal analysis using the JMI.

**Acknowledgement**: This research was supported by grant ONR N00014-96-1-0476.

## References

[1] S. Amari, A. Cichocki, and H. H. Yang. A new learning algorithm for blind signal separation. In *Advances in Neural Information Processing Systems, 8, eds. David S. Touretzky, Michael C. Mozer and Michael E. Hasselmo, MIT Press: Cambridge, MA.*, pages 757–763, 1996.

[2] G. Barrows and J. Sciortino. A mutual information measure for feature selection with application to pulse classification. In *IEEE Intern. Symposium on Time-Frequency and Time-Scale Analysis*, pages 249–253, 1996.

[3] R. Battiti. Using mutual information for selecting features in supervised neural net learning. *IEEE Trans. on Neural Networks*, 5(4):537–550, July 1994.

[4] B. Bonnlander. Nonparametric selection of input variables for connectionist learning. Technical report, PhD Thesis. University of Colorado, 1996.

[5] C. Jutten and J. Herault. Blind separation of sources, part i: An adaptive algorithm based on neuromimetic architecture. *Signal Processing*, 24:1–10, 1991.

[6] J. Moody. Prediction risk and architecture selection for neural network. In V. Cherkassky, J.H. Friedman, and H. Wechsler, editors, *From Statistics to Neural Networks: Theory and Pattern Recognition Applications*. NATO ASI Series F, Springer-Verlag, 1994.

[7] H. Pi and C. Peterson. Finding the embedding dimension and variable dependencies in time series. *Neural Computation*, 6:509–520, 1994.

[8] H. H. Yang and S. Amari. Adaptive on-line learning algorithms for blind separation: Maximum entropy and minimum mutual information. *Neural Computation*, 9(7):1457–1482, 1997.